# Bayesian model learning in human visual perception

**Gergő Orbán**
Collegium Budapest
Institute for Advanced Study
2 Szentháromság utca, Budapest,
1014 Hungary
ogergo@colbud.hu

**József Fiser**
Department of Psychology and
Volen Center for Complex Systems
Brandeis University
Waltham, Massachusetts 02454, USA
fiser@brandeis.edu

**Richard N. Aslin**
Department of Brain and Cognitive
Sciences, Center for Visual Science
University of Rochester
Rochester, New York 14627, USA
aslin@cvs.rochester.edu

**Máté Lengyel**
Gatsby Computational Neuroscience Unit
University College London
17 Queen Square, London WC1N 3AR
United Kingdom
lmate@gatsby.ucl.ac.uk

## Abstract

Humans make optimal perceptual decisions in noisy and ambiguous conditions. Computations underlying such optimal behavior have been shown to rely on probabilistic inference according to generative models whose structure is usually taken to be known a priori. We argue that Bayesian model selection is ideal for inferring similar and even more complex model structures from experience. We find in experiments that humans learn subtle statistical properties of visual scenes in a completely unsupervised manner. We show that these findings are well captured by Bayesian model learning within a class of models that seek to explain observed variables by independent hidden causes.

## 1 Introduction

There is a growing number of studies supporting the classical view of perception as probabilistic inference [1, 2]. These studies demonstrated that human observers parse sensory scenes by performing optimal estimation of the parameters of the objects involved [3, 4, 5]. Even single neurons in primary sensory cortices have receptive field properties that seem to support such a computation [6]. A core element of this Bayesian probabilistic framework is an internal model of the world, the generative model, that serves as a basis for inference. In principle, inference can be performed on several levels: the generative model can be used for inferring the values of hidden variables from observed information, but also the model itself may be inferred from previous experience [7].

Most previous studies testing the Bayesian framework in human psychophysical experiments used highly restricted generative models of perception, usually consisting of a few

observed and latent variables, of which only a limited number of parameters needed to be adjusted by experience. More importantly, the generative models considered in these studies were tailor-made to the specific pscychophysical task presented in the experiment. Thus, it remains to be shown whether more flexible, 'open-ended' generative models are used and learned by humans during perception.

Here, we use an unsupervised visual learning task to show that a general class of generative models, sigmoid belief networks (SBNs), perform similarly to humans (also reproducing paradoxical aspects of human behavior), when not only the parameters of these models but also their structure is subject to learning. Crucially, the applied Bayesian model learning embodies the Automatic Occam's Razor (AOR) effect that selects the models that are 'as simple as possible, but no simpler'. This process leads to the extraction of independent causes that efficiently and sufficiently account for sensory experience, without a pre-specification of the number or complexity of potential causes.

In section 2, we describe the experimental protocol we used in detail. Next, the mathematical framework is presented that is used to study model learning in SBNs (Section 3). In Section 4, experimental results on human performance are compared to the prediction of our Bayes-optimal model learning in the SBN framework. All the presented human experimental results were reproduced and had identical roots in our simulations: the modal model developed latent variables corresponding to the unknown underlying causes that generated the training scenes.

In Section 5, we discuss the implications of our findings. Although structure and parameter learning are not fundamentally different computations in Bayesian inference, we argue that the natural integration of these two kinds of learning lead to a behavior that accounts for human data which cannot be reproduced in some simpler alternative learning models with parameter but without structure learning. Given the recent surge of biologically plausible neural network models performing inference in belief networks we also point out challenges that our findings present for future models of probabilistic neural computations.

## 2   Experimental paradigm

Human adult subjects were trained and then tested in an unsupervised learning paradigm with a set of complex visual scenes consisting of 6 of 12 abstract unfamiliar black *shapes* arranged on a 3x3 (Exp 1) or 5x5 (Exps 2-4) white grid (Fig. 1, left panel). Unbeknownst to subjects, various subsets of the shapes were arranged into fixed spatial combinations *(combos)* (doublets, triplets, quadruplets, depending on the experiment). Whenever a combo appeared on a training scene, its constituent shapes were presented in an invariant spatial arrangement, and in no scenes elements of a combo could appear without all the other elements of the same combo also appearing. Subjects were presented with 100–200 training scenes, each scene was presented for 2 seconds with a 1-second pause between scenes. No specific instructions were given to subjects prior to training, they were only asked to pay attention to the continuous sequence of scenes.

The test phase consisted of 2AFC trials, in which two arrangements of shapes were shown sequentially in the same grid that was used in the training, and subjects were asked which of the two scenes was more familiar based on the training. One of the presented scenes was either a combo that was actually used for constructing the training set *(true combo)*, or a part of it *(embedded combo)* (e.g., a pair of adjacent shapes from a triplet or quadruplet combo). The other scene consisted of the same number of shapes as the first scene in an arrangement that might or might not have occurred during training, but was in fact a mixture of shapes from different true combos *(mixture combo)*.

Here four experiments are considered that assess various aspects of human observational

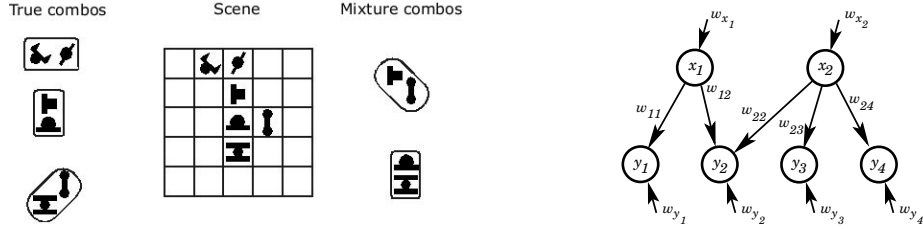

Figure 1: Experimental design (*left panel*) and explanation of graphical model parameters (*right panel*).

learning, the full set of experiments are presented elsewhere [8, 9]. Each experiment was run with 20 naïve subjects.

1. Our first goal was to establish that humans are sensitive to the statistical structure of visual experience, and use this experience for judging familiarity. In the baseline experiment 6 doublet combos were defined, three of which were presented simultaneously in any given training scene, allowing 144 possible scenes [8]. Because the doublets were not marked in any way, subjects saw only a group of random shapes arranged on a grid. The occurrence frequency of doublets and individual elements was equal across the set of scenes, allowing no obvious bias to remember any element more than others. In the test phase a true and a mixture doublet were presented sequentially in each 2AFC trial. The mixture combo was presented in a spatial position that had never appeared before.

2. In the previous experiment the elements of mixture doublets occurred together fewer times than elements of real doublets, thus a simple strategy based on tracking co-occurrence frequencies of shape-pairs would be sufficient to distinguish between them. The second, frequency-balanced experiment tested whether humans are sensitive to higher-order statistics (at least cross-correlations, which are co-occurence frequencies normalized by respective invidual occurence frequencies).

   The structure of Experiment 1 was changed so that while the 6 doublet combo architecture remained, their appearance frequency became non-uniform introducing *frequent* and *rare combos*. Frequent doublets were presented twice as often as rare ones, so that certain mixture doublets consisting of shapes from frequent doublets appeared just as often as rare doublets. Note, that the frequency of the constituent shapes of these mixture doublets was higher than that of rare doublets. The training session consisted of 212 scenes, each scene being presented twice. In the test phase, the familiarity of both single shapes and doublet combos was tested. In the doublet trials, rare combos with low appearance frequency but high correlations between elements were compared to mixed combos with higher element and equal pair appearance frequency, but lower correlations between elements.

3. The third experiment tested whether human performance in this paradigm can be fully accounted for by learning cross-correlations. Here, four triplet combos were formed and presented with equal occurrence frequencies. 112 scenes were presented twice to subjects. In the test phase two types of tests were performed. In the first type, the familiarity of a true triplet and a mixture triplet was compared, while in the second type doublets consisting of adjacent shapes embedded in a triplet combo *(embedded doublet)* were tested against mixture doublets.

4. The fourth experiment compared directly how humans treat embedded and independent (non-embedded) combos of the same spatial dimensions. Here two

quadruplet combos and two doublet combos were defined and presented with equal frequency. Each training scene consisted of six shapes, one quadruplet and one doublet. 120 such scenes were constructed. In the test phase three types of tests were performed. First, true quadruplets were compared to mixture quadruplets; next, embedded doublets were compared to mixture doublets, finally true doublets were compared to mixture doublets.

## 3 Modeling framework

The goal of Bayesian learning is to 'reverse-engineer' the generative model that could have generated the training data. Because of inherent ambiguity and stochasticity assumed by the generative model itself, the objective is to establish a *probability distribution* over possible models. Importantly, because models with parameter spaces of different dimensionality are compared, the likelihood term (Eq. 3) will prefer the simplest model (in our case, the one with fewest parameters) that can effectively account for (generate) the training data due to the AOR effect in Bayesian model comparison [7].

**Sigmoid belief networks**  The class of generative models we consider is that of two-layer sigmoid belief networks (SBNs, Fig. 1). The same modelling framework has been successfully aplied to animal learning in classical conditioning [10, 11]. The SBN architecture assumes that the state of observed binary variables ($y_j$, in our case: shapes being present or absent in a training scene) depends through a sigmoidal activation function on the state of a set of hidden binary variables ($\mathbf{x}$), which are not directly observable:

$$P\left(y_j = 1 | \mathbf{x}, \mathbf{w}_m, m\right) = \left(1 + \exp\left(-\sum_i w_{ij} x_i - w_{y_j}\right)\right)^{-1} \tag{1}$$

where $w_{ij}$ describes the (real-valued) influence of hidden variable $x_i$ on observed variable $y_j$, $w_{y_j}$ determines the spontaneous activation bias of $y_j$, and $m$ indicates the model structure, including the number of latent variables and identity of the observeds they can influence (the $w_{ij}$ weights that are allowed to have non-zero value).

Observed variables are independent conditioned on the latents (i.e. any correlation between them is assumed to be due to shared causes), and latent variables are marginally independent and have Bernoulli distributions parametrised by $\mathbf{w}_x$:

$$P\left(\mathbf{y}|\mathbf{x}, \mathbf{w}_m, m\right) = \prod_j P\left(y_j|\mathbf{x}, \mathbf{w}_m, m\right), \ P\left(\mathbf{x}|\mathbf{w}_m, m\right) = \prod_i \left(1 + \exp\left(-1^{x_i} w_{x_i}\right)\right)^{-1} \tag{2}$$

Finally, scenes ($\mathbf{y}^{(t)}$) are assumed to be iid samples from the same generative distribution, and so the probability of the training data ($\mathcal{D}$) given a specific model is:

$$P\left(\mathcal{D}|\mathbf{w}_m, m\right) = \prod_t P\left(\mathbf{y}^{(t)}|\mathbf{w}_m, m\right) = \prod_t \sum_{\mathbf{x}} \prod_j P\left(y_j^{(t)}, \mathbf{x}|\mathbf{w}_m, m\right) \tag{3}$$

The 'true' generative model that was actually used for generating training data in the experiments (Section 2) is closely related to this model, with the combos corresponding to latent variables. The main difference is that here we ignore the spatial aspects of the task, i.e. only the occurrence of a shape matters but not *where* it appears on the grid. Although in general, space is certainly not a negligible factor in vision, human behavior in the present experiments depended on the fact of shape-appearances sufficiently strongly so that this simplification did not cause major confounds in our results.

A second difference between the model and the human experiments was that in the experiments, combos were not presented completely randomly, because the number of combos

per scene was fixed (and not binomially distributed as implied by the model, Eq. 2). Nevertheless, our goal was to demonstrate the use of a general-purpose class of generative models, and although truly independent causes are rare in natural circumstances, always a fixed number of them being present is even more so. Clearly, humans are able to capture dependences between latent variables, and these should be modeled as well ([12]). Similarly, for simplicity we also ignored that subsequent scenes are rarely independent (Eq. 3) in natural vision.

**Training**  Establishing the posterior probability of any given model is straightforward using Bayes' rule:

$$P\left(\mathbf{w}_m, m | \mathcal{D}\right) \propto P\left(\mathcal{D} | \mathbf{w}_m, m\right) \ P\left(\mathbf{w}_m, m\right) \tag{4}$$

where the first term is the likelihood of the model (Eq. 3), and the second term is the prior distribution of models. Prior distributions for the weights were: $P\left(w_{ij}\right) = \mathrm{Laplace}\left(12, 2\right)$, $P\left(w_{x_i}\right) = \mathrm{Laplace}\left(0, 2\right)$, $P\left(w_{x_j}\right) = \delta\left(-6\right)$. The prior over model structure preferred simple models and was such that the distributions of the number of latents and of the number of links conditioned on the number of latents were both $\mathrm{Geometric}\left(0.1\right)$. The effect of this preference is 'washed out' with increasing training length as the likelihood term (Eq. 3) sharpens.

**Testing**  When asked to compare the familiarity of two scenes ($\mathbf{y}^A$ and $\mathbf{y}^B$) in the testing phase, the optimal strategy for subjects would be to compute the posterior probability of both scenes based on the training data

$$P\left(\mathbf{y}^Z | \mathcal{D}\right) = \sum_m \int d\mathbf{w}_m \sum_{\mathbf{x}} P\left(\mathbf{y}^Z, \mathbf{x} | \mathbf{w}_m, m\right) \ P\left(\mathbf{w}_m, m | \mathcal{D}\right) \tag{5}$$

and always (ie, with probability one) choose the one with the higher probability. However, as a phenomenological model of all kinds of possible sources of noise (sensory noise, model noise, etc) we chose a soft threshold function for computing choice probability:

$$P\left(\mathrm{choose\,A}\right) = \left(1 + \exp\left(-\beta \ \log \frac{P\left(\mathbf{y}^A | \mathcal{D}\right)}{P\left(\mathbf{y}^B | \mathcal{D}\right)}\right)\right)^{-1} \tag{6}$$

and used $\beta = 1$ ($\beta = \infty$ corresponds to the optimal strategy).

Note that when computing the probability of a test scene, we seek the probability that exactly the given scene was generated by the learned model. This means that we require not only that all the shapes that are present in the test scene are present in the generated data, but also that all the shapes that are absent from the test scene are absent from the generated data. A different scheme, in which only the presence but not the absence of the shapes need to be matched (i.e. absent observeds are marginalized out just as latents are in Eq. 5) could also be pursued, but the results of the embedding experiments (Exp. 3 and 4, see below) discourage it.

The model posterior in Eq. 4 is analytically intractable, therefore an exchange reversible-jump Markov chain Monte Carlo sampling method [10, 13, 14] was applied, that ensured fair sampling from a model space containing subspaces of differring dimensionality, and integration over this posterior in Eq. 5 was approximated by a sum over samples.

## 4   Results

Pilot studies were performed with reduced training datasets in order to test the performance of the model learning framework. First, we trained the model on data consisting of 8 observed variables ('shapes'). The 8 'shapes' were partitioned into three 'combos' of different

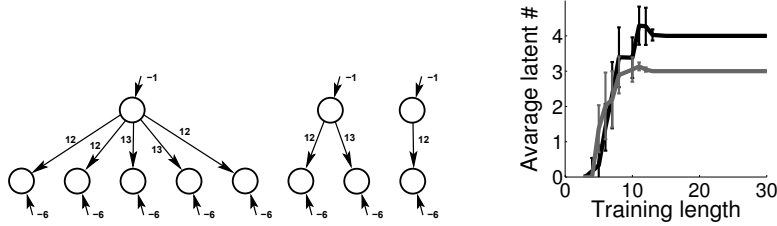

Figure 2: Bayesian learning in sigmoid belief networks. *Left panel:* MAP model of a 30-trial-long training with 8 observed variables and 3 combos. Latent variables of the MAP model reflect the relationships defined by the combos. *Right panel:* Increasing model complexity with increasing training experience. Average number of latent variables ($\pm$SD) in the model posterior distribution as a function of the length of training data was obtained by marginalizing Eq. 4 over weights $\mathbf{w}$.

sizes (5, 2, 1), two of which were presented simultaneously in each training trial. The AOR effect in Bayesian model learning should select the model structure that is of just the right complexity for describing the data. Accordingly, after 30 trials, the *maximum a posteriori* (MAP) model had three latents corresponding to the underlying 'combos' (Fig. 2, left panel). Early on in training simpler model structures dominated because of the prior preference for low latent and link numbers, but due to the simple structure of the training data the likelihood term won over in as few as 10 trials, and the model posterior converged to the true generative model (Fig. 2, right panel, gray line). Importantly, presenting more data with the same statistics did not encourage the fitting of over-complicated model structures. On the other hand, if data was generated by using more 'combos' (4 'doublets'), model learning converged to a model with a correspondingly higher number of latents (Fig. 2, right panel, black line).

In the baseline experiment (Experiment 1) human subjects were trained with six equal-sized doublet combos and were shown to recognize true doublets over mixture doublets (Fig. 3, first column). When the same training data was used to compute the choice probability in 2AFC tests with model learning, true doublets were reliably preferred over mixture doublets. Also, the MAP model showed that the discovered latent variables corresponded to the combos generating the training data (data not shown).

In Experiment 2, we sought to answer the question whether the statistical learning demonstrated in Experiment 1 was solely relying on co-occurrence frequencies, or was using something more sophisticated, such as at least cross-correlations between shapes. Bayesian model learning, as well as humans, could distinguish between rare doublet combos and mixtures from frequent doublets (Fig. 3, second column) despite their balanced co-occurrence frequencies. Furthermore, although in this comparison rare doublet combos were preferred, both humans and the model learned about the frequencies of their constituent shapes and preferred constituent single shapes of frequent doublets over those of rare doublets. Nevertheless, it should be noted that while humans showed greater preference for frequent singlets than for rare doublets our simulations predicted an opposite trend[1].

We were interested whether the performance of humans could be fully accounted for by the learning of cross-correlations, or they demonstrated more sophisticated computations.

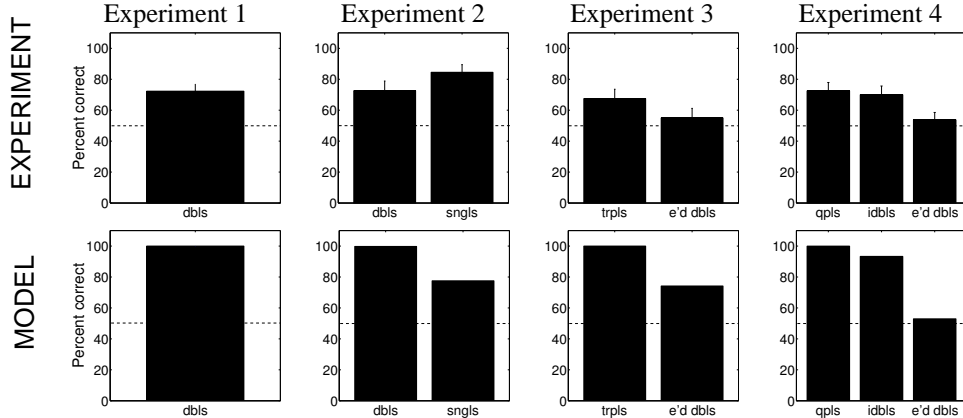

Figure 3: Comparison of human and model performance in four experiments. Bars show percent 'correct' values (choosing a true or embedded combo over a mixture combo, or a frequent singlet over a rare singlet) for human experiments (average over subjects ±SEM), and 'correct' choice probabilities (Eq. 6) for computer simulations. *Sngls:* Single shapes; *dbls*: Doublet combos; *trpls:* triplet combos; *e'd dbls:* embedded doublet combos; *qpls:* quadruple combos; *idbls:* independent doublet combos.

In Experiment 3, training data was composed of triplet combos, and beside testing true triplets against mixture triplets, we also tested embedded doublets (pairs of shapes from the same triplet) against mixture doublets (pairs of shapes from different triplets). If learning only depends on cross-correlations, we expect to see similar performance on these two types of tests. In contrast, human performace was significantly different for triplets (true triplets were preferred) and doublets (embedded and mixture doublets were not distinguished) (Fig. 3, third column). This may be seen as Gestalt effects being at work: once the 'whole' triplet is learned, its constituent parts (the embedded doublets) loose their significance. Our model reproduced this behavior and provided a straightforward explanation: latent-to-observed weights ($w_{ij}$) in the MAP model were so strong that whenever a latent was switched on it could almost only produce triplets, therefore doublets were created by spontaneous independent activation of observeds which thus produced embedded and mixture doublets with equal chance. In other words, doublets were seen as mere noise under the MAP model.

The fourth experiment tested explicitly whether embedded combos and equal-sized independent real combos are distinguished and not only size effects prevented the recognition of embedded small structures in the previous experiment. Both human experiments and Bayesian model selection demonstrated that quadruple combos as well as stand-alone doublets were reliably recognized (Fig. 3, fourth column), while embedded doublets were not.

## 5 Discussion

We demonstrated that humans flexibly yet automatically learn complex generative models in visual perception. Bayesian model learning has been implicated in several domains of high level human cognition, from causal reasoning [15] to concept learning [16]. Here we showed it being at work already at a pre-verbal stage.

We emphasized the importance of learning the *structure* of the generative model, not only its parameters, even though it is quite clear that the two cannot be formally distinguished. Nevertheless we have two good reasons to believe that structure learning is indeed impor-

tant in our case. (1) Sigmoid belief networks identical to ours but without structure learning have been shown to perform poorly on a task closely related to ours [17], Földiák's bar test [18]. More complicated models will of course be able to produce identical results, but we think our model framework has the advantage of being intuitively simple: it seeks to find the simplest possible explanation for the data assuming that it was generated by independent causes. (2) Structure learning allows Occam's automatic razor to come to play. This is computationally expensive, but together with the generative model class we use provides a neat and highly efficient way to discover 'independent components' in the data. We experienced difficulties with other models [17] developed for similar purposes when trying to reproduce our experimental findings.

Our approach is very much in the tradition that sees the finding of independent causes behind sensory data as one of the major goals of perception [2]. Although neural network models that can produce such computations exist [6, 19], none of these does model selection. Very recently, several models have been proposed for doing inference in belief networks [20, 21] but parameter learning let alone structure learning proved to be non-trivial in them. Our results highlight the importance of considering model structure learning in neural models of Bayesian inference.

### Acknowledgements

We were greatly motivated by the earlier work of Aaron Courville and Nathaniel Daw [10, 11], and hugely benefited from several useful discussions with them. We would also like to thank the insightful comments of Peter Dayan, Maneesh Sahani, Sam Roweis, and Zoltán Szatmáry on an earlier version of this work. This work was supported by IST-FET-1940 program (GO), NIH research grant HD-37082 (RNA, JF), and the Gatsby Charitable Foundation (ML).

## Footnotes

[1]This discrepancy between theory and experiments may be explained by Gestalt effects in human vision that would strongly prefer the independent processing of constituent shapes due to their clear spatial separation in the training scenes. The reconciliation of such Gestalt effects with pure statistical learning is the target of further investigations.

## References

[1] Helmholtz HLF. Treatise on Physiological Optics. New York: Dover, 1962.
[2] Barlow HB. Vision Res 30:1561, 1990.
[3] Ernst MO, Banks MS. Nature 415:429, 2002.
[4] Körding KP, Wolpert DM. Nature 427:244, 2004.
[5] Kersten D, et al. Annu Rev Psychol 55, 2004.
[6] Olshausen BA, Field DJ. Nature 381:607, 1996.
[7] MacKay DJC. Network: Comput Neural Syst 6:469, 1995.
[8] Fiser J, Aslin RN. Psych Sci 12:499, 2001.
[9] Fiser J, Aslin RN. J Exp Psychol Gen , in press.
[10] Courville AC, et al. In NIPS 16 , Cambridge, MA, 2004. MIT Press.
[11] Courville AC, et al. In NIPS 17 , Cambridge, MA, 2005. MIT Press.
[12] Hinton GE, et al. In Artificial Intelligence and Statistics , Barbados, 2005.
[13] Green PJ. Biometrika 82:711, 1995.
[14] Iba Y. Int J Mod Phys C 12:623, 2001.
[15] Tenenbaum JB, Griffiths TL. In NIPS 15 , 35, Cambridge, MA, 2003. MIT Press.
[16] Tenenbaum JB. In NIPS 11 , 59, Cambridge, MA, 1999. MIT Press.
[17] Dayan P, Zemel R. Neural Comput 7:565, 1995.
[18] Földiak P. Biol Cybern 64:165, 1990.
[19] Dayan P, et al. Neural Comput 7:1022, 1995.
[20] Rao RP. Neural Comput 16:1, 2004.
[21] Deneve S. In NIPS 17 , Cambridge, MA, 2005. MIT Press.
